# Bounds on the complexity of recurrent neural network implementations of finite state machines

**Bill G. Horne**
NEC Research Institute
4 Independence Way
Princeton, NJ 08540

**Don R. Hush**
EECE Department
University of New Mexico
Albuquerque, NM 87131

## Abstract

In this paper the efficiency of recurrent neural network implementations of $m$-state finite state machines will be explored. Specifically, it will be shown that the node complexity for the unrestricted case can be bounded above by $O(\sqrt{m})$. It will also be shown that the node complexity is $O(\sqrt{m \log m})$ when the weights and thresholds are restricted to the set $\{-1, 1\}$, and $O(m)$ when the fan-in is restricted to two. Matching lower bounds will be provided for each of these upper bounds assuming that the state of the FSM can be encoded in a subset of the nodes of size $\lceil \log m \rceil$.

## 1 Introduction

The topic of this paper is understanding how efficiently neural networks scale to large problems. Although there are many ways to measure efficiency, we shall be concerned with *node complexity*, which as its name implies, is a calculation of the required number of nodes. Node complexity is a useful measure of efficiency since the amount of resources required to implement or even simulate a recurrent neural network is typically related to the number of nodes. Node complexity can also be related to the efficiency of learning algorithms for these networks and perhaps to their generalization ability as well. We shall focus on the node complexity of recurrent neural network implementations of finite state machines (FSMs) when the nodes of the network are restricted to threshold logic units.

In the 1960s it was shown that recurrent neural networks are capable of implementing arbitrary FSMs. The first result in this area was due to Minsky [7], who showed that $m$–state FSMs can be implemented in a fully connected recurrent neural network. Although circuit complexity was not the focus of his investigation it turns out that his construction, yields $O(m)$ nodes. This construction was also guaranteed to use weight values limited to the set $\{1, 2\}$. Since a recurrent neural network with $k$ hard–limiting nodes is capable of representing as many as $2^k$ states, one might wonder if an $m$–state FSM could be implemented by a network with $\log m$ nodes. However, it was shown in [1] that the node complexity for a standard fully connected network is $\Omega\left((m \log m)^{1/3}\right)$. They were also able to improve upon Minsky's result by providing a construction which is guaranteed to yield no more than $O\left(m^{3/4}\right)$ nodes. In the same paper lower bounds on node complexity were investigated as the network was subject to restrictions on the possible range of weight values and the fan–in and fan–out of the nodes in the network. Their investigation was limited to fully connected recurrent neural networks and they discovered that the node complexity for the case where the weights are restricted to a finite size set is $\Omega\left(\sqrt{m \log m}\right)$. Alternatively, if the nodes in the network were restricted to have a constant fan–in then the node complexity becomes $\Omega(m)$. However, they left open the question of how tight these bounds are and if they apply to variations on the basic architecture. Other recent work includes investigation of the node complexity for networks with continuous valued nonlinearities [14]. However, it can also be shown that when continuous nonlinearities are used, recurrent neural networks are far more powerful than FSMs; in fact, they are Turing equivalent [13].

In this paper we improve the upper bound on the node complexity for the unrestricted case to $O(\sqrt{m})$. We also provide upper bounds that match the lower bounds above for various restrictions. Specifically, we show that a node complexity of $O\left(\sqrt{m \log m}\right)$ can be achieved if the weights are restricted to the set $\{-1, 1\}$, and that the node complexity is $O(m)$ for the case when the fan–in of each node in the network is restricted to two. Finally, we explore the possibility that implementing finite state machines in more complex models might yield a lower node complexity. Specifically, we explore the node complexity of a general recurrent neural network topology, that is capable of simulating a variety of popular recurrent neural network architectures. Except for the unrestricted case, we will show that the node complexity is no different for this architecture than for the fully connected case if the number of feedback variables is limited to $\lceil \log m \rceil$, i.e. if the state of the FSM is encoded optimally in a subset of the nodes. We leave it as an open question if a sparser encoding can lead to a more efficient implementation.

## 2  Background

### 2.1  Finite State Machines

FSMs may be defined in several ways. In this paper we shall be concerned with Mealy machines, although our approach can easily be extended to other formulations to yield equivalent results.

**Definition 1** A *Mealy machine* is a quintuple $\mathcal{M} = (Q, q_0, \Sigma, \Delta, \phi)$, where $Q$ is a finite set of states; $q_0$ is the initial state; $\Sigma$ is the input alphabet; $\Delta$ is the output alphabet; and $\phi : Q \times \Sigma \longrightarrow Q \times \Delta$ is the combined transition and output function. $\square$

Throughout this paper both the input and output alphabets will be binary (i.e. $\Sigma = \Delta = \{0, 1\}$). In general, the number of states, $m = |Q|$, may be arbitrary. Since any element of $Q$ can be encoded as a binary vector whose minimum length is $\lceil \log m \rceil$, the function $\phi$ can be implemented as a boolean logic function of the form

$$\phi : \{0, 1\}^{\lceil \log m \rceil + 1} \longrightarrow \{0, 1\}^{\lceil \log m \rceil + 1} . \tag{1}$$

The number, $N_{\mathcal{M}}$, of different minimal FSMs with $m$ states will be used to determine lower bounds on the number of gates required to implement an arbitrary FSM in a recurrent neural network. It can easily be shown that $(2m)^m \leq N_{\mathcal{M}}$ [5]. However, it will be convenient to reexpress $N_{\mathcal{M}}$ in terms of $n = \lceil \log m \rceil + 1$ as follows

$$2^{(n-1)2^{n-2}} \leq N_{\mathcal{M}}. \tag{2}$$

## 2.2   Recurrent Neural Networks

The fundamental processing unit in the models we wish to consider is the *perceptron*, which is a biased, linearly weighted sum of its inputs followed by a hard–limiting nonlinearity whose output is zero if its input is negative and one otherwise. The *fan–in* of the perceptron is defined to be the number of non–zero weights. When the values of $x_i$ are binary (as they are in this paper), the perceptron is often referred to as a *threshold logic unit* (TLU).

A count of the number of different *partially specified threshold logic functions*, which are threshold logic functions whose values are only defined over $\nu$ vertices of the unit hypercube, will be needed to develop lower bounds on the node complexity required to implement an arbitrary logic function. It has been shown that this number, denoted $L_n^{\nu}$, is [15]

$$L_n^{\nu} \leq \frac{2\nu^n}{n!}. \tag{3}$$

As pointed out in [10], many of the most popular discrete–time recurrent neural network models can be implemented as a feedforward network whose outputs are fed back recurrently through a set of unit time delays. In the most generic version of this architecture, the feed forward section is *lower triangular*, meaning the $l^{th}$ node is the only node in layer $l$ and receives input from all nodes in previous layers (including the input layer). A lower triangular network of $k$ threshold logic elements is the most general topology possible for a feedforward network since all other feedforward networks can be viewed as a special case of this network with the appropriate weights set equal to zero. The most direct implementation of this model is the architecture proposed in [11]. However, many recurrent neural network architectures can be cast into this framework. For example, fully connected networks [3] fit this model when the the feedforward network is simply a single layer of nodes. Even models which appear very different [2, 9] can be cast into this framework.

## 3   The unrestricted case

The unrestricted case is the most general, and thus explores the inherent power of recurrent neural networks. The unrestricted case is also important because it serves as a baseline from which one can evaluate the effect of various restrictions on the node complexity.

In order to derive an upper bound on the node complexity of recurrent neural network implementations of FSMs we shall utilize the following lemma, due to Lupanov [6]. The proof of this lemma involves a construction that is extremely complex and beyond the scope of this paper.

**Lemma 1 (Lupanov, 1973)** Arbitrary boolean logic functions with $x$ inputs and $y$ outputs can be implemented in a network of perceptrons with a node complexity of

$$O\left(\sqrt{\frac{y2^x}{x - \log y}}\right).$$

$\square$

**Theorem 1** Multilayer recurrent neural networks can implement FSMs having $m$ states with a node complexity of $O\left(\sqrt{m}\right)$. $\square$

**Proof:** Since an $m$–state FSM can be implemented in a recurrent neural network in which the multilayer network performs a mapping of the form in equation (1), then using $n = m = \lceil \log m \rceil + 1$, and applying Lemma 1 gives an upper bound of $O\left(\sqrt{m}\right)$. *Q.E.D.*

**Theorem 2** Multilayer recurrent neural networks can implement FSMs having $m$ states with a node complexity of $\Omega\left(\sqrt{m}\right)$ if the number of unit time delays is $\lceil \log m \rceil$. $\square$

**Proof:** In order to prove the theorem we derive an expression for the maximum number of functions that a $k$–node recurrent neural network can compute and compare that against the minimum number of finite state machines. Then we solve for $k$ in terms of the number of states of the FSM.

Specifically, we wish to manipulate the inequality

$$2^{(n-1)2^{n-2}} \leq \underbrace{n! \binom{k-1}{n-1}}_{(a)} \underbrace{\prod_{i=0}^{k-1} \frac{2^{n(n+i)+1}}{(n+i)!}}_{(b)}.$$

where the left hand side is given in equation (2), (a) represents the total number of ways to choose the outputs and feedback variables of the network, and (b) represents the total number of logic functions computable by the feed forward section of the network, which is lower triangular. Part (a) is found by simple combinatorial arguments and noting that the last node in the network must be used as either an output or feedback node. Part (b) is obtained by the following argument: If the state is optimally encoded in $\lceil \log m \rceil$ nodes, then only $\lceil \log m \rceil$ variables need

to be fed back. Together with the external input this gives $n = \lceil \log m \rceil + 1$ local inputs to the feedforward network. Repeated application of (3) with $\nu = 2^n$ yields expression (b).

Following a series of algebraic manipulations it can easily be shown that there exists a constant $c$ such that

$$n2^n \leq ck^2n.$$

Since $n = \lceil \log m \rceil + 1$ it follows that $k = \Omega\left(\sqrt{m}\right)$.                    *Q.E.D.*

## 4    Restriction on weights and thresholds

All threshold logic functions can be implemented with perceptrons whose weight and threshold values are integers. It is well known that there are threshold logic functions of $n$ variables that require a perceptron with weights whose maximum magnitude is $\Omega(2^n)$ and $O(n^{n/2})$ [8]. This implies that if a perceptron is to be implemented digitally, the number of bits required to represent each weight and threshold in the worst case will be a superlinear function of the fan–in. This is generally undesirable; it would be far better to require only a logarithmic number of bits per weight, or even better, a constant number of bits per weight. We will be primarily be interested in the most extreme case where the weights are limited to values from the set $\{-1, 1\}$.

In order to derive the node complexity for networks with weight restrictions, we shall utilize the following lemma, proved in [4].

**Lemma 2** Arbitrary boolean logic functions with $x$ inputs and $y$ outputs can be implemented in a network of perceptrons whose weights and thresholds are restricted to the set $\{-1, 1\}$ with a node complexity of $\Theta\left(\sqrt{y2^x}\right)$.                    □

This lemma is not difficult to prove, however it is beyond the scope of this paper. The basic idea involves using a decomposition of logic functions proposed in [12]. Specifically, a boolean function $f$ may always be decomposed into a disjunction of $2^r$ terms of the form $\tilde{x}_1\tilde{x}_2 \ldots \tilde{x}_r f_i(x_{r+1}, \ldots, x_n)$, one for each conjunction of the first $r$ variables, where $\tilde{x}_j$ represents either a complemented or uncomplemented version of the input variable $x_j$ and each $f_i$ is a logic function of the last $n - r$ variables. This expression can be implemented directly in a neural network. With negligible number of additional nodes, the construction can be implemented in such a way that all weights are either $-1$ or $1$. Finally, the variable $r$ is optimized to yield the minimum number of nodes in the network.

**Theorem 3** Multilayer recurrent neural networks that have nodes whose weights and thresholds are restricted to the set $\{-1, 1\}$ can implement FSMs having $m$ states with a node complexity of $O\left(\sqrt{m \log m}\right)$.                    □

**Proof:** Since an $m$–state FSM can be implemented in a recurrent neural network in which the multilayer network performs a mapping of the form in equation (1), then using $n = m = \lceil \log m \rceil + 1$, and applying Lemma 2 gives an upper bound of $O\left(\sqrt{m \log m}\right)$.                    *Q.E.D.*

**Theorem 4** Multilayer recurrent neural networks that have nodes whose weights and thresholds are restricted to a set of size $|W|$ can implement FSMs having $m$ states with a node complexity of $\Omega\left(\sqrt{\frac{m\log m}{\log|W|}}\right)$ if the number of unit time delays is $\lceil\log m\rceil$. ☐

**Proof:** The proof is similar to the proof of Theorem 2 which gave a lower bound for the node complexity required in an arbitrary network of threshold logic units. Here, the inequality we wish to manipulate is given by

$$2^{(n-1)2^{n-2}} \leq \underbrace{n!\binom{k-1}{n-1}}_{(a)} \underbrace{\prod_{i=0}^{k-1}|W|^{n+i+1}}_{(b)}.$$

where the left hand side and (a) are computed as before and (b) represents the maximum number of ways to configure the nodes in the network when there are only $|W|$ choices for each weight and threshold. Following a series of algebraic manipulations it can be shown that there exists a constant $c$ such that

$$n2^n \leq ck^2\log|W|.$$

Since $n = \lceil\log m\rceil + 1$ it follows that $k = \Omega\left(\sqrt{\frac{m\log m}{\log|W|}}\right).$     *Q.E.D.*

Clearly, for $W = \{-1,1\}$ this lower bound matches the upper bound in Theorem 3.

## 5   Restriction on fan–in

A limit on the fan–in of a perceptron is another important practical restriction. In the networks discussed so far each node has an unlimited fan–in. In fact, in the constructions described above, many nodes receive inputs from a polynomial number of nodes (in terms of $m$) in a previous layer. In practice it is not possible to build devices that have such a large connectivity. Restricting the fan–in to 2, is the most severe restriction, and will be of primary interest in this paper.

Once again, in order to derive the node complexity for restricted fan–in, we shall utilize the following lemma, proved in [4].

**Lemma 3** Arbitrary boolean logic functions with $x$ inputs and $y$ outputs can be implemented in a network of perceptrons restricted to fan–in 2 with a node complexity of

$$\Theta\left(\frac{y2^x}{x+\log y}\right).$$

☐

This proof of this lemma is very similar to the proof of Lemma 2. Here Shannon's decomposition is used with $r = 2$ to recursively decompose the logic function into a set of trees, until each tree has depth $d$. Then, all possible functions of the last $n - d$ variables are implemented in an inverted tree–like structure, which feeds into the bottom of the trees. Finally, $d$ is optimized to yield the minimum number of nodes.

**Theorem 5** Multilayer recurrent neural networks that have nodes whose fan–in is restricted to two can implement FSMs having $m$ states with a node complexity of $O(m)$                                                                       □

**Proof:** Since an $m$–state FSM can be implemented in a recurrent neural network in which the multilayer network performs a mapping of the form in equation (1), then using $n = m = \lceil \log m \rceil + 1$, and applying Lemma 3 gives an upper bound of $O(m)$.                                                                 Q.E.D.

**Theorem 6** Multilayer recurrent neural networks that have nodes whose fan–in is restricted to two can implement FSMs having $m$ states with a node complexity of $\Omega(m)$ if the number of unit time delays is $\lceil \log m \rceil$.          □

**Proof:** Once again the proof is similar to Theorem 2, which gave a lower bound for the node complexity required in an arbitrary network of threshold logic units. Here, the inequality we need to solve for is given by

$$2^{(n-1)2^{n-2}} \le \underbrace{n! \left( \begin{array}{c} k-1 \\ n-1 \end{array} \right)}_{(a)} \underbrace{\prod_{i=0}^{k-1} 14 \left( \begin{array}{c} n+i \\ 2 \end{array} \right)}_{(b)}$$

where the left hand side and (a) are computed as before and (b) represents the maximum number of ways to configure the nodes in the network. The term $\left( \begin{array}{c} n+i \\ 2 \end{array} \right)$ is used since a node in the $i^{th}$ layer has $n+i$ possible inputs from which two are chosen. The constant 14 represents the fourteen possible threshold logic functions of two variables. Following a series of algebraic manipulations it can be shown that there exists a constant $c$ such that

$$n2^n \le ck \log k$$

Since $n = \lceil \log m \rceil + 1$ it follows that $k = \Omega(m)$.                        Q.E.D.

# 6  Summary

In summary, we provide new bounds on the node complexity of implementing FSMs with recurrent neural networks. These upper bounds match lower bounds developed in [1] for fully connected recurrent networks when the size of the weight set or the fan–in of each node is finite. Although one might speculate that more complex networks might yield more efficient constructions, we showed that these lower bounds do not change for restrictions on weights or fan–in, at least when the state of the FSM is encoded optimally in a subset of $\lceil \log m \rceil$ nodes. When the network is unrestricted, this lower bound matches our upper bound. We leave it as an open question if a sparser encoding of the state variables can lead to a more efficient implementation.

One interesting aspect of this study is that there is really not much difference in efficiency when the network is totally unrestricted and when there are severe restrictions placed on the weights. Assuming that our bounds are tight, then there

is only a $\sqrt{\log m}$ penalty for restricting the weights to either $-1$ or 1. To get some idea for how marginal this difference is consider that for a finite state machine with $m = 18 \times 10^{18}$ states, $\sqrt{\log m}$ is only eight!

A more detailed version of this paper can be found in [5].

## References

[1] N. Alon, A.K. Dewdney, and T.J. Ott. Efficient simulation of finite automata by neural nets. *JACM*, 38(2):495–514, 1991.

[2] A.D. Back and A.C. Tsoi. FIR and IIR synapses, a new neural network architecture for time series modeling. *Neural Computation*, 3(3):375–385, 1991.

[3] J.J. Hopfield. Neural networks and physical systems with emergent collective computational abilities. *Proc. Nat. Acad. Sci.*, 79:2554–2558, 1982.

[4] B.G. Horne and D.R. Hush. On the node complexity of neural networks. Technical Report EECE 93–003, Dept. EECE, U. New Mexico, 1993.

[5] B.G. Horne and D.R. Hush. Bounds on the complexity of recurrent neural network implementations of finite state machines. Technical Report EECE 94–001, Dept. EECE, U. New Mexico, 1994.

[6] O.B. Lupanov. The synthesis of circuits from threshold elements. *Problemy Kibernetiki*, 26:109–140, 1973.

[7] M. Minsky. *Computation: Finite and infinite machines.* Prentice–Hall, 1967.

[8] S. Muroga. *Threshold Logic and Its Applications.* Wiley, 1971.

[9] K.S. Narendra and K. Parthasarathy. Identification and control of dynamical systems using neural networks. *IEEE Trans. on Neural Networks*, 1:4–27, 1990.

[10] O. Nerrand *et al.* Neural networks and nonlinear adaptive filtering: Unifying concepts and new algorithms. *Neural Computation*, 5(2):165–199, 1993.

[11] A.J. Robinson and F. Fallside. Static and dynamic error propagation networks with application to speech coding. In D.Z. Anderson, editor, *Neural Information Processing Systems*, pages 632–641, 1988.

[12] C. Shannon. The synthesis of two–terminal switching circuits. *Bell Sys. Tech. J.*, 28:59–98, 1949.

[13] H. Siegelmann and E.D. Sontag. Neural networks are universal computing devices. Technical Report SYCON–91–08, Rutgers Ctr. for Sys. and Cont., 1991.

[14] H.T. Siegelmann, E.D. Sontag, and C.L. Giles. The complexity of language recognition by neural networks. In *Proc. IFIP 12$^{th}$ World Comp. Cong.*, pages 329–335, 1992.

[15] R.O. Winder. Bounds on threshold gate realizability. *IEEE Trans. on Elect. Comp.*, EC–12:561–564, 1963.
